# Convergent Combinations of Reinforcement Learning with Linear Function Approximation

**Ralf Schoknecht**
ILKD
University of Karlsruhe, Germany
*ralf.schoknecht@ilkd.uni-karlsruhe.de*

**Artur Merke**
Lehrstuhl Informatik 1
University of Dortmund, Germany
*artur.merke@udo.edu*

## Abstract

Convergence for iterative reinforcement learning algorithms like TD(0) depends on the sampling strategy for the transitions. However, in practical applications it is convenient to take transition data from arbitrary sources without losing convergence. In this paper we investigate the problem of repeated synchronous updates based on a fixed set of transitions. Our main theorem yields sufficient conditions of convergence for combinations of reinforcement learning algorithms and linear function approximation. This allows to analyse if a certain reinforcement learning algorithm and a certain function approximator are *compatible*. For the combination of the residual gradient algorithm with grid-based linear interpolation we show that there exists a *universal constant* learning rate such that the iteration converges independently of the concrete transition data.

## 1 Introduction

The strongest convergence guarantees for reinforcement learning (RL) algorithms are available for the tabular case, where temporal difference algorithms for both policy evaluation and the general control problem converge with probability one independently of the concrete sampling strategy as long as all states are sampled infinitely often and the learning rate is decreased appropriately [2]. In large, possibly continuous, state spaces a tabular representation and adaptation of the value function is not feasible with respect to time and memory considerations. Therefore, linear feature-based function approximation is often used. However, it has been shown that synchronous TD(0), i.e. dynamic programming, diverges for general linear function approximation [1]. Convergence with probability one for TD($\lambda$) with general linear function approximation has been proved in [12]. They establish the crucial condition of sampling states according to the steady-state distribution of the Markov chain in order to ensure convergence. This requirement is reasonable for the pure prediction task but may be disadvantageous for policy improvement as shown in [6] because it may lead to bad action choices in rarely visited parts of the state space. When transition data is taken from arbitrary sources a certain sampling distribution cannot be assured which may prevent convergence.

An alternative to such iterative TD approaches are least-squares TD (LSTD) methods [4, 3, 6, 8]. They eliminate the learning rate parameter and carry out a matrix inversion in order to compute the fixed point of the iteration directly. In [4] a least-squares approach for TD(0) is presented which is generalised to TD($\lambda$) in [3]. Both approaches still sample the states according to the steady-state distribution. In [6, 8] arbitrary sampling distributions are used such that the transition data could be taken from any source. This may yield solutions that are not achievable by the corresponding iterative approach because this iteration diverges. All the LSTD approaches have the problem that the matrix to be inverted may be singular. This case can occur if the basis functions are not linearly independent or if the Markov chain is not recurrent. In order to apply the LSTD approach the problem would have to be preprocessed by sorting out the linear dependent basis functions and the transient states of the Markov chain. In practice one would like to save this additional work.

Thus, the least-squares TD algorithm can fail due to matrix singularity and the iterative TD(0) algorithm can fail if the sampling distribution is different from the steady-state distribution. Hence, there are problems for which neither an iterative nor a least-squares TD solution exist. The actual reason for the failure of the iterative TD(0) approach lies in an incompatible combination of the RL algorithm and the function approximator. Thus, the idea is that either a change in the RL algorithm or a change in the approximator may yield a convergent iteration. Here, a change in the TD(0) algorithm is not meant to completely alter the character of the algorithm. We require that only modifications of the TD(0) algorithm be considered that are *consistent* according to the definition in the next section.

In this paper we propose a unified framework for the analysis of a whole class of *synchronous* iterative RL algorithms combined with arbitrary linear function approximation. For the sparse iteration matrices that occur in RL such an iterative approach is superior to a method that uses matrix inversion as the LSTD approach does [5]. Our main theorem states sufficient conditions under which combinations of RL algorithms and linear function approximation converge. We hope that these conditions and the convergence analysis, that is based on the eigenvalues of the iteration matrix, bring new insight in the interplay of RL and function approximation. For an arbitrary linear function approximator and for arbitrary fixed transition data the theorem allows to predict the existence of a *constant* learning rate such that the synchronous residual gradient algorithm [1] converges. Moreover, in combination with interpolating grid-based function approximators we are able to specify a formula for a constant learning rate such that the synchronous residual gradient algorithm converges *independently* of the transition data. This is very useful because otherwise the learning rate would have to be decreased which slows down convergence.

## 2 A Framework for Synchronous Iterative RL Algorithms

For a Markov decision process (MDP) with $N$ states $S = \{s_1, \dots, s_N\}$, action space $A$, state transition probabilities $p : (S, S, A) \to [0, 1]$ and stochastic reward function $r : (S, A) \to \mathbb{R}$ policy evaluation is concerned with solving the Bellman equation

$$V^\pi = \gamma P^\pi V^\pi + R^\pi \tag{1}$$

for a fixed policy $\pi : S \to A$. $V_i^\pi$ denotes the value of state $s_i$, $P_{i,j}^\pi = p(s_i, s_j, \pi(s_i))$, $R_i^\pi = E\{r(s_i, \pi(s_i))\}$ and $\gamma$ is the discount factor. As the policy $\pi$ is fixed we will omit it in the following to make notation easier.

If the state space $S$ gets too large the exact solution of equation (1) becomes very costly with respect to both memory and computation time. Therefore, often linear

feature-based function approximation is applied. The value function $V$ is represented as a linear combination of basis functions $\{\Phi_1, \ldots, \Phi_F\}$ which can be written as $V = \Phi w$, where $w \in \mathbb{R}^F$ is the parameter vector describing the linear combination and $\Phi = (\Phi_1 | \ldots | \Phi_F) \in \mathbb{R}^{N \times F}$ is the matrix with the basis functions as columns. The rows of $\Phi$ are the feature vectors $\varphi(s_i) \in \mathbb{R}^F$ for the states $s_i$.

A popular algorithm for updating the parameter vector $w$ after a single transition $x_i \to z_i$ with reward $r_i$ is the TD(0)-algorithm [11]

$$w^{n+1} = w^n + \alpha \varphi(x_i)[r_i + \gamma \varphi(z_i)^T w^n - \varphi(x_i)^T w^n] = (I_F + \alpha A_i)w^n + \alpha b_i, \quad (2)$$

where $\alpha$ is the learning rate, $A_i = \varphi(x_i)[\gamma \varphi(z_i) - \varphi(x_i)]^T$, $b_i = \varphi(x_i)r_i$ and $I_F$ is the identity matrix in $\mathbb{R}^F$. In the following we investigate the synchronous update for a fixed set of $m$ transitions $T = \{(x_i, z_i, r_i) | i = 1, \ldots, m\}$. The start states $x_i$ are sampled with respect to the probability distribution $\rho$, the next states $z_i$ are sampled according to $p(x_i, \cdot)$ and the rewards $r_i$ are sampled from $r(x_i)$. The synchronous update for the transition set $T$ can then be written in matrix notation as

$$w^{n+1} = (I_F + \alpha A_{TD})w^n + \alpha b_{TD} \quad (3)$$

with $A_{TD} = A_1 + \ldots + A_m$ and $b_{TD} = b_1 + \ldots + b_m$. Let $X \in \mathbb{R}^{m \times N}$ with $X_{i,j} = 1$ if $x_i = s_j$ and 0 otherwise. Then, $\Phi^x = X\Phi \in \mathbb{R}^{m \times F}$ is the matrix with feature vector $\varphi(x_i)$ as its $i$-th row. Define $Z$ and $\Phi^z$ accordingly for the states $z_i$. With the vector of obtained rewards $r = (r_1, \ldots, r_m)^T$ we have $A_{TD} = (\Phi^x)^T(\gamma \Phi^z - \Phi^x)$ and $b_{TD} = (\Phi^x)^T r$.

The synchronous TD(0) algorithm is an instance of a much broader class of RL algorithms. The residual gradient algorithm [1], for example, minimises the Bellman error by gradient descent. In the following, let $\Theta = \gamma \Phi^z - \Phi^x$. The matrix $\frac{1}{m}\hat{D} = \frac{1}{m}X^T X \in \mathbb{R}^{N \times N}$ is diagonal and denotes the relative frequency of state $s_i$ as start state in the transition data $T$. Let $\check{D}$ be the diagonal matrix with the inverse entries of $\hat{D}$. For $\hat{D}_{i,i} = 0$ set $\check{D}_{i,i} = 0$. The matrix of the relative frequencies for the state transitions from $s_i$ to $s_j$ is given by $\hat{P} = \check{D}X^T Z$ and the vector of the average reward in the different states $s_i$ is given by $\hat{R} = \check{D}X^T r$. It can be shown that the weighted Bellman error for the synchronous update $E_B(w) = \frac{1}{2}\left[(\gamma\hat{P} - I_N)\Phi w + \hat{R}\right]^T \frac{1}{m}\hat{D}\left[(\gamma\hat{P} - I_N)\Phi w + \hat{R}\right]$ with the estimated entities $\hat{P}$, $\hat{R}$ and $\hat{D}$ instead of the unknown expected values $P$, $R$ and $D$ is equivalent to the expression $E_B(w) = \frac{1}{2m}\left[\Theta w + r\right]^T X\check{D}X^T\left[\Theta w + r\right]$. Thus, for the residual gradient algorithm the update rule (3) becomes $w^{n+1} = (I_F + \alpha A_{RG})w^n + \alpha b_{RG}$ with $A_{RG} = -\Theta^T X\check{D}X^T\Theta$ and $b_{RG} = -\Theta^T X\check{D}X^T r$. The synchronous TD(0) and the residual gradient algorithm can be analysed in an unified framework with $A = \Psi^T\Theta$ and $b = \Psi^T r$. By setting $\Psi_{TD} = \Phi^x$ and $\Psi_{RG} = -X\check{D}X^T\Theta$, for example, one obtains the TD(0) algorithm and the residual gradient algorithm respectively. Moreover, varying $\Psi$ yields a whole class of algorithms. We denote such algorithms as *consistent RL algorithms* if two conditions are fulfilled. First, for a tabular representation the algorithm converges to an optimal solution $w^*$ with Bellman error zero. And second, if the algorithm converges with a linear function approximator it achieves the same Bellman error independently of the initial value $w^0$. This class of RL algorithms includes the *Kaczmarz* rule [9], which is similar to the NTD(0) rule [4], or the *uniform* update rule described in [7]. In general, these algorithms yield different solutions when function approximation is used. For the TD(0) and the residual gradient algorithm this is shown in [10]. However, a general assessment of the solution quality of the different algorithms is still missing.

# 3 Convergence Results

The convergence properties of RL algorithms for synchronous updates in the general framework presented in the last section are described in the following main theorem of our paper. It generalises the case of repeated single-transition updates [7] to repeated multi-transition updates. For the following let $[M]$ be the span of the columns of a matrix $M$ and $[M]^\perp$ the orthogonal complement of $[M]$.

**Theorem 1** *Let $w^{n+1} = (I_F + \alpha A)w^n + \alpha b$ be the synchronous update rule for the transition data $T$. Let $A \in \mathbb{R}^{F \times F}$ be representable as $A = C^T D$ with some $C, D \in \mathbb{R}^{k \times F}$ and $b \in \mathbb{R}^F$ be representable as $b = C^T v$ with some $v \in \mathbb{R}^k$. Let $K = DC^T \in \mathbb{R}^{k \times k}$ and $p(x) = (-1)^k (x - \lambda_1)^{\beta_1} \cdots (x - \lambda_l)^{\beta_l}$ be the characteristic polynomial of $K$ over $\mathbb{C}$ with $|\lambda_1| > \ldots > |\lambda_l|$. Also, let $\mathcal{E}_{\lambda_i}^K$ be the eigenspace corresponding to eigenvalue $\lambda_i$ and $H = \max_i \{ \frac{|\lambda_i|^2}{|\mathrm{Re}(\lambda_i)|} \}$. If the following assumptions hold*

*(a)* $\forall i : (\mathrm{Re}(\lambda_i) < 0) \vee \lambda_i = 0$

*(b)* $\dim(\mathcal{E}_{\lambda_i}^K) = \beta_i$ *for* $\lambda_i = 0$

*(c)* $[C^T] \cap [D^T]^\perp = \{\mathbf{0}\}$

*then the limit $w^* = \lim_{n \to \infty} w^n$ exists for all learning rates $0 < \alpha < \alpha_L$, where the limit learning rate $\alpha_L$ satisfies $\alpha_L = \frac{2}{H}$. The limit $w^*$ may depend on the initial value $w^0$. Note, if the $\lambda_i$ leading to the maximum of $H$ is real then $H = |\lambda_i|$.*

A proof of this theorem can be found in the appendix. General convergence conditions of iterations have been examined in numerical mathematics. A standard result states that if the absolute value of the largest eigenvalue of the iteration matrix $I_F + \alpha A$, i.e. the spectral radius, is smaller than one, then the iteration converges to the *unique* fixed point $w^* = -A^{-1}b$ [5] (Theorem 2.1.1). In our case, however, the matrix $A$ may not be invertible. This happens, for example, if the features $\Phi_i$ in the feature matrix $\Phi$ are linearly dependent. If $A$ is not invertible it has eigenvalue zero and, thus, $I_F + \alpha A$ has eigenvalue one. Conditions (b) and (c) in the above theorem are needed in order to compensate for the singularity of $A$ and to assure convergence. If the iteration converges for singular $A$ the fixed point depends on the initial value $w^0$ and is no longer unique. Therefore, for consistent RL algorithms we require that the Bellman error of all fixed points be the same. Thus, the quality of the obtained solution to the policy evaluation problem is independent of the initial value. However, the suitability of different $w^*$ for a policy improvement step can vary but this question is not addressed here.

An important implication of Theorem 1 concerns the choice of the learning rate. If sampling were involved in the update rule the learning rate would have to be decreased in the standard manner ($\sum_t \alpha_t = \infty$, $\sum_t \alpha_t^2 < \infty$) in order to fulfil the condition for stochastic approximation algorithms. However, for a fixed set of updates and certain synchronous RL algorithms with linear feature-based function approximation Theorem 1 predicts the existence of a *constant* learning rate. In general the computation of this learning rate would require knowledge of the eigenvalues of $K$ which may not be directly available. As the following proposition shows, for certain combinations of RL algorithms and linear function approximation a universal constant learning rate exists such that the iteration in Theorem 1 converges. The proof can be found in the appendix.

**Proposition 1** *For an appropriate constant choice of the learning rate $\alpha$ the residual gradient algorithm will converge independently of the linear function approximation scheme when applied to the problem of repeated synchronous multi-transition*

*updates. The residual gradient algorithm is a consistent RL algorithm. If the residual gradient algorithm is combined with grid-based linear interpolation over an arbitrary triangulation of the state space and the transition set contains $m$ transitions then the iteration converges for all $\alpha < \frac{2}{m(1+\gamma^2)}$.*

A choice of the learning rate $\alpha < \frac{2}{H}$ according to Theorem 1 yields a convergent iteration. However, this might not be the best choice with respect to asymptotic convergence rate. The asymptotic convergence rate is better for matrices with lower spectral radius [5], which yields a criterion for the choice of an *optimal* learning rate $\alpha^*$. If $K$ has only real eigenvalues then we can deduce a particular simple formula for $\alpha^*$. Assume that all nonzero eigenvalues of $K$ satisfy $\lambda_i \in [\lambda_{\max}, \lambda_{\min}]$, where $\lambda_{\min}$ is the largest eigenvalue smaller than zero and $\lambda_{\max}$ is the eigenvalue with largest absolute value. It can be shown that the asymptotic convergence rate is determined by the eigenvalues of $I_m + \alpha K$ that are unequal one. The eigenvalues $\bar{\lambda}_i$ of $K$ are related to the eigenvalues $\bar{\lambda}_i$ of $I_m + \alpha K$ by $\bar{\lambda}_i = 1 + \alpha \lambda_i$. Hence, the interval $[\lambda_{\max}, \lambda_{\min}]$ is mapped to $[\bar{\lambda}_{\max}, \bar{\lambda}_{\min}] = [1 + \alpha \lambda_{\max}, 1 + \alpha \lambda_{\min}]$. In order to obtain a low spectral radius of $I_m + \alpha K$ this interval should lie symmetrically around zero, which is equivalent to $\bar{\lambda}_{\min} = -\bar{\lambda}_{\max}$. This yields $\alpha^* = \frac{2}{|\lambda_{\min}| + |\lambda_{\max}|} < \frac{2}{H}$ with $H = |\lambda_{\max}|$. Thus, $\alpha^*$ leads to convergence according to Theorem 1. Note also that a larger learning rate does not necessarily lead to a faster asymptotic convergence of the iteration.

## 4 Counterexample of Baird – Revisited

In this section we analyse the counterexample given by Baird in [1], and show how Theorem 1 and Proposition 1 can be applied to obtain explicit bounds for the learning rate $\alpha$ and the discount factor $\gamma$ for which the residual gradient and TD(0) algorithms converge. The matrices $\Phi$, $X$ and $Z$ are given by

$$\Phi = \begin{pmatrix} 1\,2\,0\,0\,0\,0\,0\,0 \\ 1\,0\,2\,0\,0\,0\,0\,0 \\ 1\,0\,0\,2\,0\,0\,0\,0 \\ 1\,0\,0\,0\,2\,0\,0\,0 \\ 1\,0\,0\,0\,0\,2\,0\,0 \\ 1\,0\,0\,0\,0\,0\,2\,0 \\ 2\,0\,0\,0\,0\,0\,0\,1 \end{pmatrix}, \quad X = \begin{pmatrix} 1\,0\,0\,0\,0\,0\,0 \\ 0\,1\,0\,0\,0\,0\,0 \\ 0\,0\,1\,0\,0\,0\,0 \\ 0\,0\,0\,1\,0\,0\,0 \\ 0\,0\,0\,0\,1\,0\,0 \\ 0\,0\,0\,0\,0\,1\,0 \\ 0\,0\,0\,0\,0\,0\,1 \end{pmatrix}, \quad Z = \begin{pmatrix} 0\,0\,0\,0\,0\,0\,1 \\ 0\,0\,0\,0\,0\,0\,1 \\ 0\,0\,0\,0\,0\,0\,1 \\ 0\,0\,0\,0\,0\,0\,1 \\ 0\,0\,0\,0\,0\,0\,1 \\ 0\,0\,0\,0\,0\,0\,1 \\ 0\,0\,0\,0\,0\,0\,1 \end{pmatrix},$$

which corresponds to the synchronous update of every state transition. In the residual gradient case we have $K_{RG} = -(\gamma Z - X)\Phi((\gamma Z - X)\Phi)^T$ which has just negative eigenvalues $\sigma_{RG} = \{-4, \frac{1}{2}[-15 + 34\gamma - 35\gamma^2 \pm \sqrt{2102\gamma^2 - 812\gamma - 2380\gamma^3 + 121 + 1225\gamma^4}]\}$. Using Theorem 1 and Proposition 1 we can find a constant learning rate $\alpha$, such that the iteration converges for every $\gamma \in [0,1)$. For example, for $\gamma = 0.9$ the eigenvalues of $K_{RG}$ are $\sigma_{RG} = \{-0.0204, -4, -12.7296\}$ and Theorem 1 yields $\alpha < 0.1571$ which is also almost equal to the optimal learning rate $\alpha^* \approx 0.1569$.

In the TD(0) case we have to analyse the matrix $K_{TD} = -(\gamma Z - X)\Phi(X\Phi)^T$, which has the eigenvalues $\sigma_{TD} = \{-4, \frac{1}{2}[-15 + 17\gamma \pm \sqrt{289\gamma^2 - 406\gamma + 121}]\}$. There are eigenvalues of $K_{TD}$ with positive real part for $\gamma \geqslant 0.89$. In such cases we have divergence for every $\alpha > 0$ as described in [1] for $\gamma = 0.9$. However, contradicting the argument in [1] the TD(0) algorithm converges for all $\gamma \leqslant 0.88$ if the learning rate is chosen appropriately. For example, for $\gamma = 0.4$ all eigenvalues are negative ($\sigma_{TD} = \{-3.0, -4, -5.2\}$), so condition (a) and (b) of Theorem 1 are trivially fulfilled. Condition (c) can also be shown by simple computation, and therefore using Theorem 1 we obtain convergence for $\alpha < 0.384$ and optimal asymptotic convergence for $\alpha^* \approx 0.244$, which is much smaller.

# 5  Conclusions

For the problem of repeated synchronous updates based on a fixed set of transitions we have proved sufficient conditions of convergence for arbitrary combinations of reinforcement learning algorithms and linear function approximation. Our main theorem yields a rule for determining a problem dependent learning rate such that the algorithm converges. For a combination of the residual gradient algorithm with grid-based linear interpolation we have deduced a constant learning rate such that the algorithm converges independently of the concrete transition data. Moreover, we have derived a general formula for an optimal learning rate with respect to asymptotic convergence. Finally we have applied our main theorem to fully analyse the example Baird gives for the divergence of TD(0) [1].

## Appendix

**Lemma 1** *Let $D$ be a real $m \times F$ matrix and $C^T$ a real $F \times m$ matrix, where $m > F$. Then $K = DC^T$ has the same eigenvalues as $A = C^T D$ and additionally the eigenvalue zero with multiplicity $(F - m)$. Let $\mathcal{H}_\lambda^K$ be the generalised eigenspace of $K$ corresponding to the eigenvalue $\lambda$ and $\mathcal{H}_\lambda^A$ the generalised eigenspace of $A$ corresponding to the eigenvalue $\lambda$. Then, $C^T \mathcal{H}_\lambda^K \subseteq \mathcal{H}_\lambda^A$ and $D \mathcal{H}_\lambda^A \subseteq \mathcal{H}_\lambda^K$. For $\lambda \neq 0$ it even holds that $C^T \mathcal{H}_\lambda^K = \mathcal{H}_\lambda^A$ and $D \mathcal{H}_\lambda^A = \mathcal{H}_\lambda^K$.*

**Proof:**  The generalised eigenspace $\mathcal{H}_\lambda^K$ has index $s_\lambda^K$ if $s_\lambda^K$ is the smallest number for which $\ker(K - \lambda I_m)^{s_\lambda^K} = \ker(K - \lambda I_m)^{s_\lambda^K + 1}$ holds, where $I_k$ denotes the identity in $\mathbb{R}^{k \times k}$. Let $x \in \mathcal{H}_\lambda^K$, i.e. $(K - \lambda I_m)^{s_\lambda^K} x = 0$. With $C^T K^i = A^i C^T$ we have

$$C^T (K - \lambda I_m)^{s_\lambda^K} x = C^T \Big( \sum_{i=0}^{s_\lambda^K} \binom{s_\lambda^K}{i} K^i \lambda^{s_\lambda^K - i} \Big) x = (A - \lambda I_F)^{s_\lambda^K} C^T x. \qquad (4)$$

Thus, $C^T x \in \mathcal{H}_\lambda^A$. And with the same argument we obtain $Dx \in \mathcal{H}_\lambda^K$ from $x \in \mathcal{H}_\lambda^A$. Therefore, $C^T \mathcal{H}_\lambda^K \subseteq \mathcal{H}_\lambda^A$ and $D \mathcal{H}_\lambda^A \subseteq \mathcal{H}_\lambda^K$ Let $\lambda \neq 0$ and $B_\lambda^K$ a basis in $\mathcal{H}_\lambda^K$. As the Jordan block of $K$ corresponding to $\mathcal{H}_\lambda^K$ is invertible the vectors $C^T B_\lambda^K$ are linearly independent and therefore form a basis of the span $[C^T B_\lambda^K]$. With the above consideration we have $[C^T B_\lambda^K] \subseteq \mathcal{H}_\lambda^A$. If this is a real subset $C^T B_\lambda^K$ can be completed to form a basis $B_\lambda^A$ of $\mathcal{H}_\lambda^A$ with $|B_\lambda^K| < |B_\lambda^A|$. Then we have that $D B_\lambda^A$ is linearly independent and $[D B_\lambda^A] \subseteq \mathcal{H}_\lambda^K$. Moreover, we have $\dim(\mathcal{H}_\lambda^K) = |B_\lambda^K| < |B_\lambda^A| = \dim([D B_\lambda^A]) \leqslant \dim(\mathcal{H}_\lambda^K)$, which is a contradiction. Therefore, $C^T \mathcal{H}_\lambda^K = [C^T B_\lambda^K] = \mathcal{H}_\lambda^A$. Similarly, we obtain $D \mathcal{H}_\lambda^A = \mathcal{H}_\lambda^K$. Thus, the multiplicities of the eigenvalues $\lambda \neq 0$ of $A$ and $K$ are the same. The multiplicity of the eigenvalue zero of matrix $K$ is by $(F - m)$ larger than that of matrix $A$.  $\square$

**Proof of Theorem 1:**  Due to assumption (a) and Lemma 1 every eigenvalue of $A$ is either zero or has a real part less than zero. If the real part of every eigenvalue of $A$ is less than zero, $A$ is invertible. For invertible matrices Theorem 2.1.1 from [5] states that the iteration converges if and only if the spectral radius $\varrho(I_F + \alpha A)$, i.e. the largest eigenvalue, is less than 1. For every eigenvalue $\lambda_i$ of $A$ obviously $1 + \alpha \lambda_i$ is an eigenvalue of $I_F + \alpha A$. With $H = \max_i \{ \frac{|\lambda_i|^2}{|\mathrm{Re}(\lambda_i)|} \}$ we obtain for $\alpha > 0$

$$\varrho(I_F + \alpha A) < 1 \iff \forall i : |1 + \alpha \lambda_i| < 1 \iff \alpha < \frac{2}{H}. \qquad (5)$$

This completes the proof if all eigenvalues of $A$ have a negative real part.

In the following let $A$ have the eigenvalue $\lambda_1 = 0$. The vector space $\mathbb{R}^F$ can be represented as the direct sum of the generalised eigenspaces $\mathbb{R}^F = \mathcal{H}_0^A \oplus \mathcal{H}_{\lambda_2}^A \oplus$

$\cdots \oplus \mathcal{H}^A_{\lambda_l}$. In the following we write $\bar{\mathcal{H}}^A_0 = \mathcal{H}^A_{\lambda_2} \oplus \cdots \oplus \mathcal{H}^A_{\lambda_l}$ because this is a complementary space of $\mathcal{H}^A_0$. As the generalised eigenspaces of $A$ are invariant against $A$, i.e. $\forall x \in \mathcal{H}^A_{\lambda_i} : Ax \in \mathcal{H}^A_{\lambda_i}$, the iteration $w^{n+1} = (I_F + \alpha A)w^n + \alpha b$ can be decomposed in two parts, one in the generalised eigenspace $\mathcal{H}^A_0$ and the other in the complementary space $\bar{\mathcal{H}}^A_0$. Let $w^n = \tilde{w}^n + \bar{w}^n$ and $b = \tilde{b} + \bar{b}$, where $\tilde{w}^n, \tilde{b} \in \mathcal{H}^A_0$ and $\bar{w}^n, \bar{b} \in \bar{\mathcal{H}}^A_0$. Then we have

$$w^{n+1} = w^n + \alpha(Aw^n + b) = \underbrace{\tilde{w}^n + \alpha(A\tilde{w}^n + \tilde{b})}_{\tilde{w}^{n+1}\in\mathcal{H}^A_0} + \underbrace{\bar{w}^n + \alpha(A\bar{w}^n + \bar{b})}_{\bar{w}^{n+1}\in\bar{\mathcal{H}}^A_0} \quad (6)$$

Thus, the convergence analysis can be carried out separately for the two iterations. The matrix $A$ in iteration $\bar{w}^{n+1} = \bar{w}^n + \alpha(A\bar{w}^n + \bar{b})$ is not invertible. However, the iteration takes place in the subspace $\bar{\mathcal{H}}^A_0$. In this subspace the mapping associated with $A$ is invertible. Therefore, $A$ can be replaced by an invertible matrix $\bar{A}$ that does not alter the iteration in $\bar{\mathcal{H}}^A_0$. The matrix $\bar{A}$ can be constructed such that $\varrho(I_F + \alpha\bar{A}) = \varrho(I_F + \alpha A)$. Therefore, according to the considerations above the iteration converges for $0 < \alpha < \frac{2}{H}$.

In the following we show that the iteration in $\mathcal{H}^A_0$ is the identity and therefore trivially converges. According to assumption (b) $\mathcal{H}^K_0 = \mathcal{E}^K_0$. All $v \in \mathbb{R}^m$ can be represented as $v = \tilde{v} + \bar{v}$ with $\tilde{v} \in \mathcal{E}^K_0$ and $\bar{v} \in \bar{\mathcal{H}}^K_0 = \mathcal{H}^K_{\lambda_2} \oplus \cdots \oplus \mathcal{H}^K_{\lambda_l}$. According to Lemma 1 $C^T\bar{\mathcal{H}}^K_0 = \bar{\mathcal{H}}^A_0$ and $C^T\mathcal{H}^K_0 \subseteq \mathcal{H}^A_0$ hold. Therefore, for $\tilde{b} + \bar{b} = b = C^Tv$ we have $\tilde{b} = C^T\tilde{v}$ and $\bar{b} = C^T\bar{v}$. Let $\mathcal{E}^K_0 \neq \{\mathbf{0}\}$. Then, for all $\tilde{v} \in \mathcal{E}^K_0$

$$\mathbf{0} = K\tilde{v} = DC^T\tilde{v} \Longrightarrow C^T\tilde{v} \in [C^T] \cap [D^T]^\perp \overset{(c)}{\Longrightarrow} C^T\tilde{v} = \mathbf{0}.$$

For $\mathcal{E}^K_0 = \{\mathbf{0}\}$ we also obtain $C^T\tilde{v} = \mathbf{0}$ because $\tilde{v} = \mathbf{0}$. Therefore, we have $C^T\mathcal{E}^K_0 = \{\mathbf{0}\}$ and, as a consequence, $\tilde{b} = C^T\tilde{v} = \mathbf{0}$. The last that remains to show is that $A\tilde{w} = \mathbf{0}$ for all $\tilde{w} \in \mathcal{H}^A_0$. According to Lemma 1 we know that $D\tilde{w} \in \mathcal{H}^K_0$. Assumption (b) says that $\mathcal{H}^K_0 = \mathcal{E}^K_0$ and from the above considerations we know that $C^T\mathcal{E}^K_0 = \{\mathbf{0}\}$. Therefore, $A\tilde{w} = C^T(D\tilde{w}) = \mathbf{0}$. Thus, the iteration in $\mathcal{H}^A_0$ is the identity. As both parts of the iteration converge the overall iteration also converges which completes that part of the proof.

The limit $\bar{w}^*$ of $\bar{w}^{n+1} = \bar{w}^n + \alpha(\bar{A}\bar{w}^n + \bar{b})$ is unique and we have $\bar{w}^* = \bar{A}^{-1}\bar{b}$. The limit of $\tilde{w}^{n+1} = \tilde{w}^n + \alpha(A\tilde{w}^n + \tilde{b})$ is not unique, but depends on the initial value $\tilde{w}^0$. It holds that $\tilde{w}^* = \tilde{w}^0$. Therefore, the limit $w^* = \tilde{w}^* + \bar{w}^*$ depends on the initial value $w^0$.

**Proof of Proposition 1:** For the residual gradient algorithm we have $A_{RG} = -\Theta^T X \check{D} X^T \Theta$ and $b_{RG} = -\Theta^T X \check{D} X^T r$. In order to apply Theorem 1 this is decomposed in $A_{RG} = C^T D$ and $b_{RG} = C^T v$ with $C = -D = \sqrt{\check{D}} X^T \Theta$ and $v = -\sqrt{\check{D}} X^T r$. As the diagonal entries of $\check{D}$ are positive we can write $\sqrt{\check{D}}$ for the diagonal matrix whose entries are the square roots of $\check{D}$. Thus $[C^T] = [D^T]$ which yields condition (c) of Theorem 1. Moreover, the matrix $K = DC^T = -CC^T$ is symmetric and therefore diagonalisable. Hence, condition (b) is fulfilled and all eigenvalues are real. Let now $\lambda \neq 0$ be an eigenvalue of $K$ and let $x$ be a corresponding eigenvector. Then $0 > -(C^Tx)^T(C^Tx) = x^TKx = \lambda x^Tx$ which yields $\lambda < 0$. Thus, all requirements are fulfilled and for an appropriate choice of $\alpha$ the residual gradient algorithm converges independently of the concrete form of the function approximation scheme.

The consistency of the residual gradient algorithm can be shown formally but due to space limitations we only give the following informal proof. The algorithm minimises

the Bellman error, which is a quadratic objective function. Hence, there are no local optima and if the global optimum is not unique, the values of all global optima are identical. Due to its gradient descent property the residual gradient algorithm converges to such a global optimum independently of the initial value. In case of a tabular representation a global minimum has Bellman error zero and corresponds to an optimal solution. Thus, the residual gradient algorithm is consistent.

A detailed description of how grid-based linear interpolation works in combination with RL can be found in [7]. Important for us is that in a $d$-dimensional grid each feature vector $\varphi(x)$ satisfies $0 \leqslant \varphi_i(x) \leqslant 1$ and $\sum_{i=1}^{F} \varphi_i(x) = 1$. With $\langle \cdot, \cdot \rangle$ denoting the standard scalar product and $|| \cdot ||_2$ denoting the corresponding euclidean norm, we have $|K_{i,j}| = |\langle (C^T)_i, (C^T)_j \rangle| \leqslant \max_l \{ ||(C^T)_l||_2^2 \} = \sum_{j=1}^{F} C_{l,j}^2$. According to the definition $C_{l,j} = \left( \sqrt{\breve{D}} \right)_{l,l} \sum_{k=1}^{m} X_{k,l} (\gamma \varphi_j(z_k) - \varphi_j(x_k))$ holds. Moreover, from $\hat{D} = X^T X$ it follows that $\hat{D}_{l,l} = \sum_{k=1}^{m} X_{k,l}^2 = \sum_{k=1}^{m} X_{k,l}$ because $X_{k,l}$ is either zero or one. And besides that we have $\breve{D}_{l,l} \hat{D}_{l,l} = 1$. Altogether we obtain

$$|K_{i,j}| \leqslant \gamma^2 \left( \breve{D}_{l,l} \sum_{k=1}^{m} X_{k,l} \sum_{j=1}^{F} \varphi_j(z_k) \right)^2 + \left( \breve{D}_{l,l} \sum_{k=1}^{m} X_{k,l} \sum_{j=1}^{F} \varphi_j(x_k) \right)^2 = \gamma^2 + 1.$$

It is well known that the spectral radius $\varrho$ of the matrix $K$ satisfies $\varrho(K) \leqslant ||K||$ for every norm $|| \cdot ||$. Then, for the maximum norm of $K$ we obtain $||K||_\infty = \max_{1 \leqslant i \leqslant m} \sum_{j=1}^{m} |K_{i,j}| \leqslant m(1 + \gamma^2)$. With $H = m(1 + \gamma^2)$ this yields $\varrho(K) \leqslant ||K||_\infty \leqslant H$. Thus we have a bound for the absolute value of the largest eigenvalue of $K$. According to Theorem 1 the iteration converges for $\alpha < \frac{2}{H}$. $\qquad \square$

# References

[1] L. C. Baird. Residual algorithms: Reinforcement learning with function approximation. *Proc. of the Twelfth International Conference on Machine Learning*, 1995.

[2] D. P. Bertsekas and J. N. Tsitsiklis. *Neuro Dynamic Programming*. Athena Scientific, Belmont, Massachusetts, 1996.

[3] J.A. Boyan. Least-squares temporal difference learning. In *Proceeding of the Sixteenth International Conference on Machine Learning*, pages 49–56, 1999.

[4] S.J Bradtke and A.G. Barto. Linear least-squares algorithms for temporal difference learning. *Machine Learning*, 22:33–57, 1996.

[5] A. Greenbaum. *Iterative Methods for Solving Linear Systems*. SIAM, 1997.

[6] D. Koller and R. Parr. Policy iteration for factored mdps. In *Proc. of the Sixteenth Conference on Uncertainty in Artificial Intelligence (UAI)*, pages 326–334, 2000.

[7] A. Merke and R. Schoknecht. A necessary condition of convergence for reinforcement learning with function approximation. In *Proceedings of the Nineteenth International Conference on Machine Learning*, pages 411–418, Sydney, Australia, 2002.

[8] M. G. Lagoudakis and R. Parr. Model-free least-squares policy iteration. In *Advances in Neural Information Processing Systems*, volume 14, 2002.

[9] S. Pareigis. Adaptive choice of grid and time in reinforcement learning. *Advances in Neural Information Processing Systems*, 1998.

[10] R. Schoknecht. Optimality of reinforcement learning algorithms with linear function approximation. In *Advances in Neural Information Processing Systems*, volume 15, 2003.

[11] R. S. Sutton. Learning to predict by the methods of temporal differences. *Machine Learning*, 3:9–44, 1988.

[12] J. N. Tsitsiklis and B. Van Roy. An analysis of temporal-difference learning with function approximation. *IEEE Transactions on Automatic Control*, 1997.
